# On Parallel Versus Serial Processing: A Computational Study of Visual Search

**Eyal Cohen**
Department of Psychology
Tel-Aviv University Tel Aviv 69978, Israel
eyalc@devil.tau.ac.il

**Eytan Ruppin**
Departments of Computer Science & Physiology
Tel-Aviv University Tel Aviv 69978, Israel
ruppin@math.tau.ac.il

## Abstract

A novel neural network model of pre-attention processing in visual-search tasks is presented. Using displays of line orientations taken from Wolfe's experiments [1992], we study the hypothesis that the distinction between parallel versus serial processes arises from the availability of global information in the internal representations of the visual scene. The model operates in two phases. First, the visual displays are compressed via principal-component-analysis. Second, the compressed data is processed by a target detector module in order to identify the existence of a target in the display. Our main finding is that targets in displays which were found experimentally to be processed in parallel can be detected by the system, while targets in experimentally-serial displays cannot. This fundamental difference is explained via variance analysis of the compressed representations, providing a numerical criterion distinguishing parallel from serial displays. Our model yields a mapping of response-time slopes that is similar to Duncan and Humphreys's "search surface" [1989], providing an explicit formulation of their intuitive notion of feature similarity. It presents a neural realization of the processing that may underlie the classical metaphorical explanations of visual search.

# 1    Introduction

This paper presents a neural-model of pre-attentive visual processing. The model explains why certain displays can be processed very fast, "in parallel", while others require slower, "serial" processing, in subsequent attentional systems. Our approach stems from the observation that the visual environment is overflowing with diverse information, but the biological information-processing systems analyzing it have a limited capacity [1]. This apparent mismatch suggests that *data compression* should be performed at an early stage of perception, and that via an accompanying process of *dimension reduction*, only a few essential features of the visual display should be retained. We propose that only parallel displays incorporate global features that enable fast target detection, and hence they can be processed pre-attentively, with all items (target and distractors) examined at once. On the other hand, in serial displays' representations, global information is obscure and target detection requires a serial, attentional scan of local features across the display. Using principal-component-analysis (PCA), our main goal is to demonstrate that neural systems employing compressed, dimensionally reduced representations of the visual information can successfully process only parallel displays and not serial ones. The source of this difference will be explained via variance analysis of the displays' projections on the principal axes.

The modeling of visual attention in cognitive psychology involves the use of metaphors, e.g., Posner's beam of attention [2]. A visual attention system of a surviving organism must supply fast answers to burning issues such as detecting a target in the visual field and characterizing its primary features. An attentional system employing a constant-speed beam of attention [3] probably cannot perform such tasks fast enough and a pre-attentive system is required. Treisman's feature integration theory (FIT) describes such a system [4]. According to FIT, features of separate dimensions (shape, color, orientation) are first coded pre-attentively in a locations map and in separate feature maps, each map representing the values of a particular dimension. Then, in the second stage, attention "glues" the features together conjoining them into objects at their specified locations. This hypothesis was supported using the *visual-search* paradigm [4], in which subjects are asked to detect a target within an array of distractors, which differ on given physical dimensions such as color, shape or orientation. As long as the target is significantly different from the distractors in one dimension, the reaction time (RT) is short and shows almost no dependence on the number of distractors (low RT slope). This result suggests that in this case the target is detected pre-attentively, in parallel. However, if the target and distractors are similar, or the target specifications are more complex, reaction time grows considerably as a function of the number of distractors [5, 6], suggesting that the displays' items are scanned serially using an attentional process.

FIT and other related cognitive models of visual search are formulated on the conceptual level and do not offer a detailed description of the processes involved in transforming the visual scene from an ordered set of data points into given values in specified feature maps. This paper presents a novel computational explanation of the source of the distinction between parallel and serial processing, progressing from general metaphorical terms to a neural network realization. Interestingly, we also come out with a computational interpretation of some of these metaphorical terms, such as feature similarity.

## 2   The Model

We focus our study on visual-search experiments of line orientations performed by Wolfe et. al. [7], using three set-sizes composed of 4, 8 and 12 items. The number of items equals the number of distractors + target in target displays, and in non-target displays the target was replaced by another distractor, keeping a constant set-size. Five experimental conditions were simulated: (A) - a 20 degrees tilted target among vertical distractors (homogeneous background). (B) - a vertical target among 20 degrees tilted distractors (homogeneous background). (C) - a vertical target among heterogeneous background ( a mixture of lines with ±20, ±40 , ±60 , ±80 degrees orientations). (E) - a vertical target among two flanking distractor orientations (at ±20 degrees), and (G) - a vertical target among two flanking distractor orientations (±40 degrees). The response times (RT) as a function of the set-size measured by Wolfe et. al. [7] show that type A, B and G displays are scanned in a parallel manner (1.2, 1.8, 4.8 msec/item for the RT slopes), while type C and E displays are scanned serially (19.7, 17.5 msec/item). The input displays of our system were prepared following Wolfe's prescription: Nine images of the basic line orientations were produced as nine matrices of gray-level values. Displays for the various conditions of Wolfe's experiments were produced by randomly assigning these matrices into a 4x4 array, yielding 128x100 display-matrices that were transformed into 12800 display-vectors. A total number of 2400 displays were produced in 30 groups (80 displays in each group): 5 conditions (A, B, C, E, G ) × target/non-target × 3 set-sizes (4, 8, 12).

Our model is composed of two neural network modules connected in sequence as illustrated in Figure 1: a PCA module which compresses the visual data into a set of principal axes, and a Target Detector (TD) module. The latter module uses the compressed data obtained by the former module to detect a target within an array of distractors. The system is presented with line-orientation displays as described above.

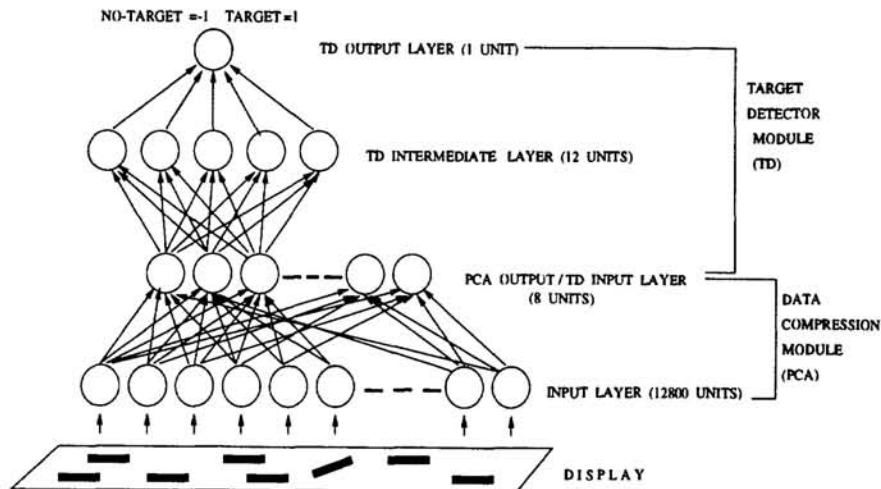

Figure 1: General architecture of the model

For the PCA module we use the neural network proposed by Sanger, with the connections' values updated in accordance with his Generalized Hebbian Algorithm (GHA) [8]. The outputs of the trained system are the projections of the display-vectors along the first few principal axes, ordered with respect to their eigenvalue magnitudes. Compressing the data is achieved by choosing outputs from the first

few neurons (maximal variance and minimal information loss). Target detection in our system is performed by a feed-forward (FF) 3-layered network, trained via a standard back-propagation algorithm in a supervised-learning manner. The input layer of the FF network is composed of the first eight output neurons of the PCA module. The transfer function used in the intermediate and output layers is the hyperbolic tangent function.

## 3  Results

### 3.1  Target Detection

The performance of the system was examined in two simulation experiments. In the first, the PCA module was trained only with "parallel" task displays, and in the second, only with "serial" task displays. There is an inherent difference in the ability of the model to detect targets in parallel versus serial displays. In parallel task conditions (A, B, G) the target detector module learns the task after a comparatively small number (800 to 2000) of epochs, reaching performance level of almost 100%. However, the target detector module is not capable of learning to detect a target in serial displays (C, E conditions). Interestingly, these results hold (1) whether the preceding PCA module was trained to perform data compression using parallel task displays or serial ones, (2) whether the target detector was a linear simple perceptron, or the more powerful, non-linear network depicted in Figure 1, and (3) whether the full set of 144 principal axes (with non-zero eigenvalues) was used.

### 3.2  Information Span

To analyze the differences between parallel and serial tasks we examined the eigenvalues obtained from the PCA of the training-set displays. The eigenvalues of condition B (parallel) displays in 4 and 12 set-sizes and of condition C (serial-task) displays are presented in Figure 2. Each training set contains a mixture of target and non-target displays.

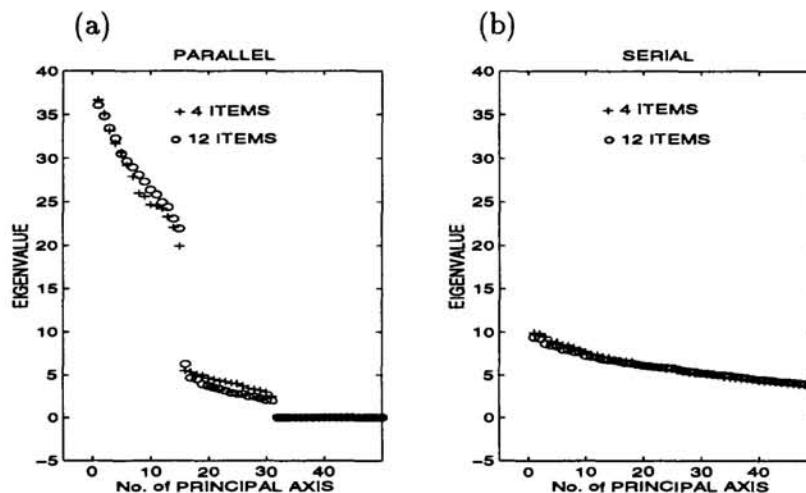

Figure 2: Eigenvalues spectrum of displays with different set-sizes, for parallel and serial tasks. Due to the sparseness of the displays (a few black lines on white background), it takes only 31 principal axes to describe the parallel training-set in full (see fig 2a. Note that the remaining axes have zero eigenvalues, indicating that they contain no additional information.), and 144 axes for the serial set (only the first 50 axes are shown in fig 2b).

As evident, the eigenvalues distributions of the two display types are fundamentally different: in the parallel task, most of the eigenvalues "mass" is concentrated in the first few (15) principal axes, testifying that indeed, the dimension of the parallel displays space is quite confined. But for the serial task, the eigenvalues are distributed almost uniformly over 144 axes. This inherent difference is independent of set-size: 4 and 12-item displays have practically the same eigenvalue spectra.

## 3.3 Variance Analysis

The target detector inputs are the projections of the display-vectors along the first few principal axes. Thus, some insight to the source of the difference between parallel and serial tasks can be gained performing a variance analysis on these projections. The five different task conditions were analyzed separately, taking a group of 85 target displays and a group of 85 non-target displays for each set-size. Two types of variances were calculated for the projections on the 5th principal axis: The "within groups" variance, which is a measure of the statistical noise within each group of 85 displays, and the "between groups" variance, which measures the separation between target and non-target groups of displays for each set-size. These variances were averaged for each task (condition), over all set-sizes. The resulting ratios Q of within-groups to between-groups standard deviations are: $Q_A = 0.0259$, $Q_B = 0.0587$, and $Q_G = 0.0114$ for parallel displays (A, B, G), and $Q_E = 0.2125$ $Q_C = 0.771$ for serial ones (E, C).

As evident, for parallel task displays the Q values are smaller by an order of magnitude compared with the serial displays, indicating a better separation between target and non-target displays in parallel tasks. Moreover, using Q as a criterion for parallel/serial distinction one can predict that displays with $Q << 1$ will be processed in parallel, and serially otherwise, in accordance with the experimental response time (RT) slopes measured by Wolfe et. al. [7]. This differences are further demonstrated in Figure 3, depicting projections of display-vectors on the sub-space spanned by the 5, 6 and 7th principal axes. Clearly, for the parallel task (condition B), the PCA representations of the target-displays (plus signs) are separated from non-target representations (circles), while for serial displays (condition C) there is no such separation. It should be emphasized that there is no other principal axis along which such a separation is manifested for serial displays.

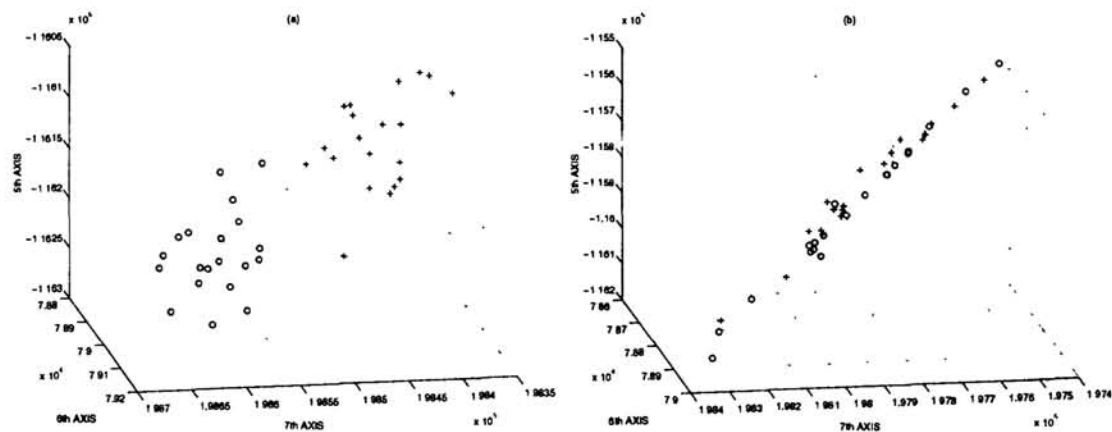

Figure 3: Projections of display-vectors on the sub-space spanned by the 5, 6 and 7th principal axes. Plus signs and circles denote target and non-target display-vectors respectively, (a) for a parallel task (condition B), and (b) for a serial task (condition C). Set-size is 8 items.

While Treisman and her co-workers view the distinction between parallel and serial tasks as a fundamental one, Duncan and Humphreys [5] claim that there is no sharp distinction between them, and that search efficiency varies continuously across tasks and conditions. The determining factors according to Duncan and Humphreys are the similarities between the target and the non-targets (T-N similarities) and the similarities between the non-targets themselves (N-N similarity). Displays with homogeneous background (high N-N similarity) and a target which is significantly different from the distractors (low T-N similarity) will exhibit parallel, low RT slopes, and vice versa. This claim was illustrated by them using a qualitative "search surface" description as shown in figure 4a. Based on results from our variance analysis, we can now examine this claim quantitatively: We have constructed a "search surface", using actual numerical data of RT slopes from Wolfe's experiments, replacing the N-N similarity axis by its mathematical manifestation, the within-groups standard deviation, and N-T similarity by between-groups standard deviation [1]. The resulting surface (Figure 4b) is qualitatively similar to Duncan and Humphreys's. This interesting result testifies that the PCA representation succeeds in producing a viable realization of such intuitive terms as inputs similarity, and is compatible with the way we perceive the world in visual search tasks.

(a)                                                       (b)

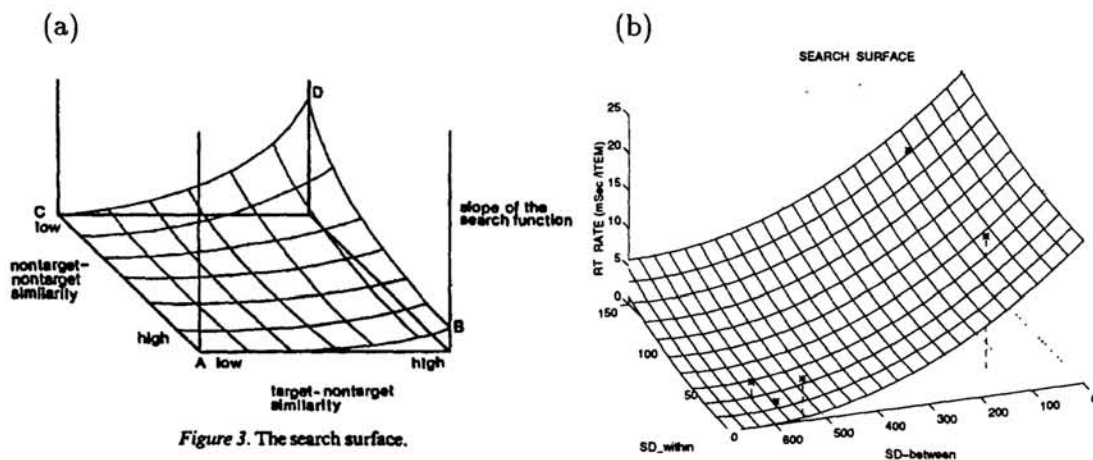

*Figure 3.* The search surface.

Figure 4: RT rates versus: (a) Input similarities (the search surface, reprinted from Duncan and Humphreys, 1989). (b) Standard deviations (within and between) of the PCA variance analysis. The asterisks denote Wolfe's experimental data.

## 4   Summary

In this work we present a two-component neural network model of pre-attentional visual processing. The model has been applied to the visual search paradigm performed by Wolfe et. al. Our main finding is that when global-feature compression is applied to visual displays, there is an inherent difference between the representations of serial and parallel-task displays: The neural network studied in this paper has succeeded in detecting a target among distractors only for displays that were experimentally found to be processed in parallel. Based on the outcome of the

variance analysis performed on the PCA representations of the visual displays, we present a quantitative criterion enabling one to distinguish between serial and parallel displays. Furthermore, the resulting 'search-surface' generated by the PCA components is in close correspondence with the metaphorical description of Duncan and Humphreys.

The network demonstrates an interesting generalization ability: Naturally, it can learn to detect a target in parallel displays from examples of such displays. However, it can also learn to perform this task from examples of serial displays only! On the other hand, we find that it is impossible to learn serial tasks, irrespective of the combination of parallel and serial displays that are presented to the network during the training phase. This generalization ability is manifested not only during the learning phase, but also during the performance phase; displays belonging to the same task have a similar eigenvalue spectrum, irrespective of the actual set-size of the displays, and this result holds true for parallel as well as for serial displays.

The role of PCA in perception was previously investigated by Cottrell [9], designing a neural network which performed tasks as face identification and gender discrimination. One might argue that PCA, being a global component analysis is not compatible with the existence of local feature detectors (e.g. orientation detectors) in the cortex. Our work is in line with recent proposals [10] that there exist two pathways for sensory input processing: A fast sub-cortical pathway that contains limited information, and a slow cortical pathway which is capable of providing richer representations of the stimuli. Given this assumption this paper has presented the first neural realization of the processing that may underline the classical metaphorical explanations involved in visual search.

## Footnotes

[1] In general, each principal axis contains information from different features, which may mask the information concerning the existence of a target. Hence, the first principal axis may not be the best choice for a discrimination task. In our simulations, the 5th axis for example, was primarily dedicated to target information, and was hence used for the variance analysis (obviously, the neural network uses information from all the first eight principal axes).

# References

[1] J. K. Tsotsos. Analyzing vision at the complexity level. *Behavioral and Brain Sciences*, 13:423–469, 1990.

[2] M. I. Posner, C. R. Snyder, and B. J. Davidson. Attention and the detection of signals. *Journal of Experimental Psychology: General*, 109:160–174, 1980.

[3] Y. Tsal. Movement of attention across the visual field. *Journal of Experimental Psychology: Human Perception and Performance*, 9:523–530, 1983.

[4] A. Treisman and G. Gelade. A feature integration theory of attention. *Cognitive Psychology*, 12:97–136, 1980.

[5] J. Duncan and G. Humphreys. Visual search and stimulus similarity. *Psychological Review*, 96:433–458, 1989.

[6] A. Treisman and S. Gormican. Feature analysis in early vision: Evidence from search assymetries. *Psychological Review*, 95:15–48, 1988.

[7] J. M. Wolfe, S. R. Friedman-Hill, M. I. Stewart, and K. M. O'Connell. The role of categorization in visual search for orientation. *Journal of Experimental Psychology: Human Perception and Performance*, 18:34–49, 1992.

[8] T. D. Sanger. Optimal unsupervised learning in a single-layer linear feedforward neural network. *Neural Network*, 2:459–473, 1989.

[9] G. W. Cottrell. Extracting features from faces using compression networks: Face, identity, emotion and gender recognition using holons. *Proceedings of the 1990 Connectionist Models Summer School*, pages 328–337, 1990.

[10] J. L. Armony, D. Servan-Schreiber, J. D. Cohen, and J. E. LeDoux. Computational modeling of emotion: exploration through the anatomy and physiology of fear conditioning. *Trends in Cognitive Sciences*, 1(1):28–34, 1997.